# Kernel Dependency Estimation

**Jason Weston, Olivier Chapelle, André Elisseeff,
Bernhard Schölkopf and Vladimir Vapnik\***
Max Planck Institute for Biological Cybernetics, 72076 Tübingen, Germany
*NEC Research Institute, Princeton, NJ 08540 USA

## Abstract

We consider the learning problem of finding a dependency between a general class of objects and another, possibly different, general class of objects. The objects can be for example: vectors, images, strings, trees or graphs. Such a task is made possible by employing similarity measures in both input *and* output spaces using kernel functions, thus embedding the objects into vector spaces. We experimentally validate our approach on several tasks: mapping strings to strings, pattern recognition, and reconstruction from partial images.

## 1 Introduction

In this article we consider the rather general learning problem of finding a dependency between inputs $\mathbf{x} \in \mathcal{X}$ and outputs $\mathbf{y} \in \mathcal{Y}$ given a training set $(\mathbf{x}_1, \mathbf{y}_1), \ldots, (\mathbf{x}_m, \mathbf{y}_m) \in \mathcal{X} \times \mathcal{Y}$ where $\mathcal{X}$ and $\mathcal{Y}$ are nonempty sets. This includes conventional pattern recognition and regression estimation. It also encompasses more complex dependency estimation tasks, e.g mapping of a certain class of strings to a certain class of graphs (as in text parsing) or the mapping of text descriptions to images. In this setting, we define learning as estimating the function $f(\mathbf{x}, \alpha^*)$ from the set of functions $\{f(\cdot, \alpha), \alpha \in \Lambda\}$ which provides the minimum value of the risk function

$$R(\alpha) = \int_{\mathcal{X} \times \mathcal{Y}} L(\mathbf{y}, f(\mathbf{x}, \alpha)) dP(\mathbf{x}, \mathbf{y}) \qquad (1)$$

where $P$ is the (unknown) joint distribution of $\mathbf{x}$ and $\mathbf{y}$ and $L(\mathbf{y}, \boldsymbol{\eta})$ is a loss function, a measure of distance between the estimate $\boldsymbol{\eta}$ and the true output $\mathbf{y}$ at a point $\mathbf{x}$. Hence in this setting one is given *a priori* knowledge of the similarity measure used in the space $\mathcal{Y}$ in the form of a loss function. In pattern recognition this is often the zero-one loss, in regression often squared loss is chosen. However, for other types of outputs, for example if one was required to learn a mapping to images, or to a mixture of drugs (a drug cocktail) to prescribe to a patient then more complex costs would apply. We would like to be able to encode these costs into the method of estimation we choose.

The framework we attempt to address is rather general. Few algorithms have been constructed which can work in such a domain - in fact the only algorithm that we are aware of is *k*-nearest neighbors. Most algorithms have focussed on the pattern

recognition and regression problems and cannot deal with more general outputs. Conversely, specialist algorithms have been made for structured outputs, for example the ones of text classification which calculate parse trees for natural language sentences, however these algorithms are specialized for their tasks. Recently, kernel methods [12, 11] have been extended to deal with inputs that are structured objects such as strings or trees by linearly embedding the objects using the so-called kernel trick [5, 7]. These objects are then used in pattern recognition or regression domains. In this article we show how to construct a general algorithm for dealing with dependencies between both general inputs and general outputs. The algorithm ends up in an formulation which has a kernel function for the inputs and a kernel function (which will correspond to choosing a particular loss function) for the outputs. This also enables us (in principle) to encode specific prior information about the outputs (such as special cost functions and/or invariances) in an elegant way, although this is not experimentally validated in this work.

The paper is organized as follows. In Section 2 it is shown how to use kernel functions to measure similarity between outputs as well as inputs. This leads to the derivation of the Kernel Dependency Estimation (KDE) algorithm in Section 3. Section 4 validates the method experimentally and Section 5 concludes.

## 2 Loss functions and kernels

An informal way of looking at the learning problem consists of the following. Generalization occurs when, given a previously unseen $\mathbf{x} \in \mathcal{X}$, we find a suitable $\mathbf{y} \in \mathcal{Y}$ such that $(\mathbf{x}, \mathbf{y})$ should be "similar" to $(\mathbf{x}_1, \mathbf{y}_1), \ldots, (\mathbf{x}_m, \mathbf{y}_m)$. For outputs one is usually given a loss function for measuring similarity (this can be, but is not always, inherent to the problem domain). For inputs, one way of measuring similarity is by using a *kernel* function. A kernel $k$ is a symmetric function which is an inner product in some Hilbert space $\mathcal{F}$, i.e., there exists a map $\Phi_k : \mathcal{X} \to \mathcal{F}$ such that $k(\mathbf{x}, \mathbf{x}') = (\Phi_k(\mathbf{x}) \cdot \Phi_k(\mathbf{x}'))$. We can think of the patterns as $\Phi_k(\mathbf{x}), \Phi_k(\mathbf{x}')$, and carry out geometric algorithms in the inner product space ("feature space") $\mathcal{F}$. Many successful algorithms are now based on this approach, see e.g [12, 11]. Typical kernel functions are polynomials $k(\mathbf{x}, \mathbf{x}') = (\mathbf{x} \cdot \mathbf{x}' + 1)^p$ and RBFs $k(\mathbf{x}, \mathbf{x}') = \exp(- \|\mathbf{x} - \mathbf{x}'\|^2 / 2\sigma^2)$ although many other types (including ones which take into account prior information about the learning problem) exist.

Note that, like distances between examples in input space, it is also possible to think of the loss function as a distance measure in output space, we will denote this space $\mathcal{L}$. We can measure inner products in this space using a kernel function. We will denote this as $\ell(\mathbf{y}, \mathbf{y}') = (\Phi_\ell(\mathbf{y}) \cdot \Phi_\ell(\mathbf{y}'))$, where $\Phi_\ell : \mathcal{Y} \to \mathcal{L}$. This map makes it possible to consider a large class of nonlinear loss functions.[1] As in the traditional kernel trick for the inputs, the nonlinearity is only taken into account when computing the kernel matrix. The rest of the training is "simple" (e.g., a convex program, or methods of linear algebra such as matrix diagonalization). It also makes it possible to consider structured objects as outputs such as the ones described in [5]: strings, trees, graphs and so forth. One embeds the output objects in the space $\mathcal{L}$ using a kernel.

Let us define some kernel functions for output spaces.

In $M$-class pattern recognition, given $\mathcal{Y} = \{1, \ldots, M\}$, one often uses the distance $L(\mathbf{y}, \mathbf{y}') = 1 - [\mathbf{y} = \mathbf{y}']$, where $[\mathbf{y} = \mathbf{y}']$ is 1 if $\mathbf{y} = \mathbf{y}'$ and 0 otherwise. To construct a corresponding inner product it is necessary to embed this distance into a Euclidean space, which can be done using the following kernel:

$$\ell_{pat}(\mathbf{y}, \mathbf{y}') = \frac{1}{2}[\mathbf{y} = \mathbf{y}'], \qquad (2)$$

as $L(\mathbf{y}, \mathbf{y}')^2 = \|\Phi_\ell(\mathbf{y}) - \Phi_\ell(\mathbf{y}')\|^2 = \ell(\mathbf{y}, \mathbf{y}) + \ell(\mathbf{y}', \mathbf{y}') - 2\ell(\mathbf{y}, \mathbf{y}') = 1 - [\mathbf{y} = \mathbf{y}']$. It corresponds to embedding into a $M$-dimensional Euclidean space via the map $\Phi_\ell(\mathbf{y}) = (0, 0, \ldots, \frac{\sqrt{2}}{2}, \ldots, 0)$ where the $\mathbf{y}^{th}$ coordinate is nonzero. It is also possible to describe multi-label classification (where any one example belongs to an arbitrary subset of the $M$ classes) in a similar way.

For regression estimation, one can use the usual inner product

$$\ell_{reg}(\mathbf{y}, \mathbf{y}') = (\mathbf{y} \cdot \mathbf{y}'). \qquad (3)$$

For outputs such as strings and other structured objects we require the corresponding string kernels and kernels for structured objects [5, 7]. We give one example here, the string subsequence kernel employed in [7] for text categorization. This kernel is an inner product in a feature space consisting of all ordered subsequences of length $r$, denoted $\Sigma^r$. The subsequences, which do not have to be contiguous, are weighted by an exponentially decaying factor $\lambda$ of their full length in the text:

$$\ell(\mathbf{s}, \mathbf{t}) = \sum_{\mathbf{u} \in \Sigma^r} \psi_{\mathbf{u}}(\mathbf{s}) \cdot \psi_{\mathbf{u}}(\mathbf{t}) = \sum_{\mathbf{u} \in \Sigma^r} \sum_{\mathbf{i} : \mathbf{u} = \mathbf{s}[\mathbf{i}]} \lambda^{l(\mathbf{i})} \sum_{\mathbf{j} : \mathbf{u} = \mathbf{t}[\mathbf{j}]} \lambda^{l(\mathbf{j})} \qquad (4)$$

where $\mathbf{u} = \mathbf{x}[\mathbf{i}]$ denotes $\mathbf{u}$ is the subsequence of $\mathbf{x}$ with indices $1 \leq i_1 \leq \cdots \leq i_{|\mathbf{u}|}$ and $l(\mathbf{i}) = i_{|\mathbf{u}|} - i_1 + 1$. A fast way to compute this kernel is described in [7].

Sometimes, one would also like apply the loss given by an (arbitrary) distance matrix $\mathbf{D}$ of the loss between training examples, i.e where $\mathbf{D}_{ij} = L(\mathbf{y}_i, \mathbf{y}_j)$. In general it is not always obvious to find an embedding of such data in an Euclidian space (in order to apply kernels). However, one such method is to compute the inner product with [11, Proposition 2.27]:

$$\ell(\mathbf{y}_i, \mathbf{y}_j) = \frac{1}{2}\left( |\mathbf{D}_{ij}|^2 - \sum_{p=1}^{m} c_p |\mathbf{D}_{ip}|^2 - \sum_{q=1}^{m} c_q |\mathbf{D}_{qj}|^2 + \sum_{p,q=1}^{m} c_p c_q |\mathbf{D}_{pq}|^2 \right) \qquad (5)$$

where coefficients $c_i$ satisfy $\sum_i c_i = 1$ (e.g using $c_i = \frac{1}{m}$ for all $i$ — this amounts to using the centre of mass as an origin) . See also [3] for ways of dealing with problems of embedding distances when equation (5) will not suffice.

## 3 Algorithm

Now we will describe the algorithm for performing KDE. We wish to minimize the risk function (1) using the feature space $\mathcal{F}$ induced by the kernel $k$ and the loss function measured in the space $\mathcal{L}$ induced by the kernel $\ell$. To do this we must learn the mapping from $\Phi_k(\mathbf{x})$ to $\Phi_\ell(\mathbf{y})$. Our solution is the following: decompose $\Phi_\ell(\mathbf{y})$ into $p$ orthogonal directions using kernel principal components analysis (KPCA) (see, e.g [11, Chapter 14]). One can then learn the mapping from $\Phi_k(\mathbf{x})$ to each direction independently using a standard kernel regression method, e.g SVM regression [12] or kernel ridge regression [9]. Finally, to output an estimate $\mathbf{y}$ given a test example $\mathbf{x}$ one must solve a pre-image problem as the solution of the algorithm is initially a solution in the space $\mathcal{L}$. We will now describe each step in detail.

**1) Decomposition of outputs** Let us construct the kernel matrix $\mathbf{L}$ on the training data such that $\mathbf{L}_{ij} = \ell(\mathbf{y}_i, \mathbf{y}_j)$, and perform kernel principal components analysis on $\mathbf{L}$. This can be achieved by centering the data in feature space using: $\mathbf{L}' = (\mathbf{I} - \frac{1}{m}\mathbf{1}_m\mathbf{1}_m^\top)\mathbf{L}(\mathbf{I} - \frac{1}{m}\mathbf{1}_m\mathbf{1}_m^\top)$, where $\mathbf{I}$ is the $m$-dimensional identity matrix and $\mathbf{1}_m$ is an $m$ dimensional vector of ones. One then solves the eigenvalue problem $\lambda\boldsymbol{\alpha} = \mathbf{L}'\boldsymbol{\alpha}$ where $\boldsymbol{\alpha}^n$ is the $n^{th}$ eigenvector of $\mathbf{L}'$ which we normalize such that $1 = (\boldsymbol{\alpha}^n \cdot \mathbf{L}'\boldsymbol{\alpha}^n) = \lambda_n(\boldsymbol{\alpha}^n \cdot \boldsymbol{\alpha}^n)$. We can then compute the projection of $\Phi_\ell(\mathbf{y})$ onto the $n^{th}$ principal component $\mathbf{v}^n = \sum_{i=1}^m \alpha_i^n\Phi_\ell(\mathbf{y}_i)$ by $(\mathbf{v}^n \cdot \Phi_\ell(\mathbf{y})) = \sum_{i=1}^m \alpha_i^n\ell(\mathbf{y}_i, \mathbf{y})$.

**2) Learning the map** We can now learn the map from $\Phi_k(\mathbf{x})$ to $((\mathbf{v}^1 \cdot \Phi_\ell(\mathbf{y})), \ldots, (\mathbf{v}^p \cdot \Phi_\ell(\mathbf{y})))$ where $p$ is the number of principal components. One can learn the map by estimating each output independently. In our experiments we use kernel ridge regression [9], note that this requires only a single matrix inversion to learn all $p$ directions. That is, we minimize with respect to $\mathbf{w}$ the function $\frac{1}{m}\sum_{i=1}^m(\mathbf{y}_i - (\mathbf{w} \cdot \Phi_k(\mathbf{x}_i)))^2 + \gamma\|\mathbf{w}\|^2$ in its dual form. We thus learn each output direction $(\mathbf{v}^n \cdot \Phi_\ell(\mathbf{y}))$ using the kernel matrix $\mathbf{K}_{ij} = k(\mathbf{x}_i, \mathbf{x}_j)$ and the training labels $\hat{\mathbf{y}}_i^n = (\mathbf{v}^n \cdot \Phi_\ell(\mathbf{y}_i))$, with estimator $f_n(\mathbf{x})$:

$$f_n(\mathbf{x}) = \sum_{i=1}^m \beta_i^n k(\mathbf{x}_i, \mathbf{x}), \quad \boldsymbol{\beta}^n = (\mathbf{K} + \gamma\mathbf{I})^{-1}\hat{\mathbf{y}}^n. \tag{6}$$

**3) Solving the pre-image problem** During the testing phase, to obtain the estimate $\mathbf{y}$ for a given $\mathbf{x}$ it is now necessary to find the pre-image of the given output $\Phi_\ell(\mathbf{y})$. This can be achieved by finding:

$$\mathbf{y}(\mathbf{x}) = \operatorname{argmin}_{\mathbf{y} \in \mathcal{y}}\| ((\mathbf{v}^1 \cdot \Phi_\ell(\mathbf{y})), \ldots, (\mathbf{v}^p \cdot \Phi_\ell(\mathbf{y}))) - (f_1(\mathbf{x}), \ldots, f_p(\mathbf{x}))\|$$

For the kernel (3) it is possible to compute the solution explicitly. For other problems searching from a set of candidate solutions may be enough, e.g from the set of training set outputs $\mathbf{y}_1, \ldots, \mathbf{y}_m$; in our experiments we use this set. When more accurate solutions are required, several algorithms exist for finding approximate pre-images e.g via fixed-point iteration methods, see [10] or [11, Chapter 18] for an overview.

For the simple case of vectorial outputs with linear kernel (3), if the output is only one dimension the method of KDE boils down to the same solution as using ridge regression since the matrix $\mathbf{L}$ is rank 1 in this case. However, when there are $d$ outputs, the rank of $\mathbf{L}$ is $d$ and the method trains ridge regression $d$ times, but the kernel PCA step first decorrelates the outputs. Thus, in the special case of multiple outputs regression with a linear kernel, the method is also related to the work of [2] (see [4, page 73] for an overview of other multiple output regression methods.) In the case of classification, the method is related to Kernel Fisher Discriminant Analysis (KFD) [8].

## 4    Experiments

In the following we validate our method with several experiments. In the experiments we chose the parameters of KDE to be from the following: $\sigma^* = \{10^{-3}, 10^{-2}, 10^{-1}, 10^0, 10^1, 10^2, 10^3\}$ where $\sigma = \frac{1}{\sqrt{2\sigma^*}}$, and the ridge parameter $\gamma = \{10^{-4}, 10^{-3}, 10^{-2}, 10^{-1}, 10^0, 10^1\}$. We chose them by five fold cross validation.

## 4.1 Mapping from strings to strings

**Toy problem.** Three classes of strings consist of letters from the same alphabet of 4 letters (a,b,c,d), and strings from all classes are generated with a random length between 10 to 15. Strings from the first class are generated by a model where transitions from any letter to any other letter are equally likely. The output is the string *abad*, corrupted with the following noise. There is a probability of 0.3 of a random insertion of a random letter, and a probability of 0.15 of two random insertions. After the potential insertions there is a probability of 0.3 of a random deletion, and a probability of 0.15 of two random deletions. In the second class, transitions from one letter to itself (so the next letter is the same as the last) have probability 0.7, and all other transitions have probability 0.1. The output is the string *dbbd*, but corrupted with the same noise as for class one. In the third class only the letters *c* and *d* are used; transitions from one letter to itself have probability 0.7. The output is the string *aabc*, but corrupted with the same noise as for class one. For classes one and two any starting letter is equally likely, for the third class only *c* and *d* are (equally probable) starting letters.

| input string | | output string |
|---|---|---|
| ccdddddddd | → | aabc |
| dccccdddcd | → | abc |
| adddccccccccc | → | bb |
| bbcdcdadbad | → | aebad |
| cdaaccadcbccdd | → | abad |

Figure 1: Five examples from our artificial task (mapping strings to strings).

The task is to predict the output string given the input string. Note that this is almost like a classification problem with three classes, apart from the noise on the outputs. This construction was employed so we can also calculate classification error as a sanity check. We use the string subsequence kernel (4) from [7] for both inputs and outputs, normalized such that $k(\mathbf{x}, \mathbf{x}') = k(\mathbf{x}, \mathbf{x}')/(\sqrt{k(\mathbf{x}, \mathbf{x})}\sqrt{k(\mathbf{x}', \mathbf{x}')})$. We chose the parameters $r = 3$ and $\lambda = 0.01$. In the space induced by the input kernel $k$ we then chose a further nonlinear map using an RBF kernel: $\exp(-(k(\mathbf{x}, \mathbf{x}) + k(\mathbf{x}', \mathbf{x}') - 2k(\mathbf{x}, \mathbf{x}'))/2\sigma^2)$.

We generated 200 such strings and measured the success by calculating the mean and standard error of the loss (computed via the output kernel) over 4 fold cross validation. We chose $\sigma$ (the width of the RBF kernel) and $\gamma$ (the ridge parameter) on each trial via a further level of 5 fold cross validation. We compare our method to an adaptation of $k$-nearest neighbors for general outputs: if $k = 1$ it returns the output of the nearest neighbor, otherwise it returns the linear combination (in the space of outputs) of the $k$ nearest neighbors (in input space). In the case of $k > 1$, as well as for KDE, we find a pre-image by finding the closest training example output to the given solution. We choose $k$ again via a further level of 5 fold cross validation. The mean results, and their standard errors, are given in Table 1.

| | KDE | $k$-NN |
|---|---|---|
| string loss | $0.676 \pm 0.030$ | $0.985 \pm 0.029$ |
| classification loss | $0.125 \pm 0.012$ | $0.205 \pm 0.026$ |

Table 1: Performance of KDE and $k$-NN on the string to string mapping problem.

## 4.2 Multi-class classification problem

We next tried a multi-class classification problem, a simple special case of the general dependency estimation problem. We performed 5-fold cross validation on 1000 digits (the first 100 examples of each digit) of the USPS handwritten 16x16 pixel digit database, training with a single fold (200 examples) and testing on the remainder. We used an RBF kernel for the inputs and the zero-one multi-class classification loss for the outputs using kernel (2). We again compared to $k$-NN and also to 1-vs-rest Support Vector Machines (SVMs) (see, e.g [11, Section 7.6]). We found $k$ for $k$-NN and $\sigma$ and $\gamma$ for the other methods (we employed a ridge also to the SVM method, reulting in a squared error penalization term) by another level of 5-fold cross validation. The results are given in Table 2. SVMs and KDE give similar results (this is not too surprising since KDE gives a rather similar solution to KFD, whose similarity to SVMs in terms of performance has been shown before [8]). Both SVM and KDE outperform $k$-NN.

|  | KDE | 1-vs-rest SVM | $k$-NN |
|---|---|---|---|
| classification loss | $0.0798 \pm 0.0067$ | $0.0847 \pm 0.0064$ | $0.1250 \pm 0.0075$ |

Table 2: Performance of KDE, 1-vs-rest SVMs and $k$-NN on a classification problem of handwritten digits.

## 4.3 Image reconstruction

We then considered a problem of image reconstruction: given the top half (the first 8 pixel lines) of a USPS postal digit, it is required to estimate what the bottom half will be (we thus ignored the original labels of the data).[2] The loss function we choose for the outputs is induced by an RBF kernel. The reason for this is that a penalty that is only linear in $\mathbf{y}$ would encourage the algorithm to choose images that are "inbetween" clearly readable digits. Hence, the difficulty in this task is both choosing a good loss function (to reflect the end user's objectives) as well as an accurate estimator. We chose the width $\sigma'$ of the output RBF kernel which maximized the kernel alignment [1] with a target kernel generated via k-means clustering. We chose k=30 clusters and the target kernel is $\mathbf{K}_{ij} = 1$ if $\mathbf{x}_i$ and $\mathbf{x}_j$ are in the same cluster, and 0 otherwise. Kernel alignment is then calculated via: $A(\mathbf{K}_1, \mathbf{K}_2) = \langle \mathbf{K}_1, \mathbf{K}_2 \rangle_F / \sqrt{\langle \mathbf{K}_1, \mathbf{K}_1 \rangle_F \langle \mathbf{K}_2, \mathbf{K}_2 \rangle_F}$ where $\langle \mathbf{K}, \mathbf{K}' \rangle_F = \sum_{i,j=1}^m \mathbf{K}_{ij} \mathbf{K}'_{ij}$ is the Frobenius dot product, which gave $\sigma' = 0.35$. For the inputs we use an RBF kernel of width $\sigma$.

We again performed 5-fold cross validation on the first 1000 digits of the USPS handwritten 16x16 pixel digit database, training with a single fold (200 examples) and testing on the remainder, comparing KDE to $k$-NN and a Hopfield net.[3] The Hopfield network we used was the one of [6] implemented in the Neural Network Toolbox for Matlab. It is a generalization of standard Hopfield nets that has a nonlinear transfer function and can thus deal with scalars between -1 and +1; after building the network based on the (complete) digits of the training set we present the top half of test digits and fill the bottom half with zeros, and then find the networks equilibrium point. We then chose as output the pre-image from the training data that is closest to this solution (thus the possible outputs are the

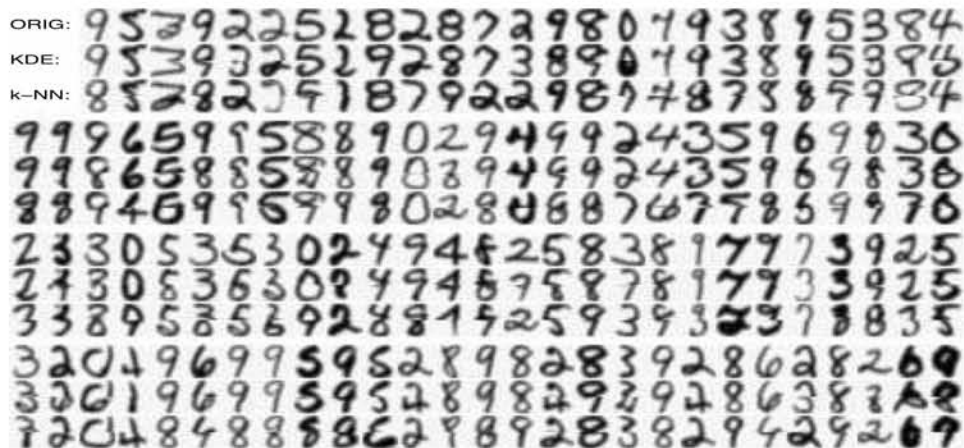

Figure 2: Errors in the digit database image reconstruction problem. Images have to be estimated using only the top half (first 8 rows of pixels) of the original image (top row) by KDE (middle row) and $k$-NN (bottom row). We show all the test examples on the first fold of cross validation where $k$-NN makes an error in estimating the correct digit whilst KDE does not (73 mistakes) and vice-versa (23 mistakes). We chose them by viewing the complete results by eye (and are thus somewhat subjective). The complete results can be found at http://www.kyb.tuebingen.mpg.de/bs/people/weston/kde/kde.html.

same as the competing algorithms). We found $\sigma$ and $\gamma$ for KDE and $k$ for $k$-NN by another level of 5-fold cross validation. The results are given in Table 3.

|  | KDE | $k$-NN | Hopfield net |
|---|---|---|---|
| RBF loss | $0.8384 \pm 0.0077$ | $0.8960 \pm 0.0052$ | $1.2190 \pm 0.0072$ |

Table 3: Performance of KDE, $k$-NN and a Hopfield network on an image reconstruction problem of handwritten digits.

KDE outperforms $k$-NN and Hopfield nets on average, see Figure 2 for comparison with $k$-NN. Note that we cannot easily compare classification rates on this problem using the pre-images selected since KDE outputs are not correlated well with the labels. For example it will use the bottom stalk of a digit "7" or a digit "9" equally if they are identical, whereas $k$-NN will not: in the region of the input space which is the top half of "9"s it will only output the bottom half of "9"s. This explains why measuring the class of the pre-images compared to the true class as a classification problem yields a lower loss for $k$-NN, $0.2345 \pm 0.0058$, compared to KDE, $0.2985 \pm 0.0147$ and Hopfield nets, $0.5910 \pm 0.0137$. Note that if we performed classification as in Section 4.2 but using only the first 8 pixel rows then $k$-NN yields $0.2345 \pm 0.0058$, but KDE yields $0.1878 \pm 0.0098$ and 1-vs-rest SVMs yield $0.1942 \pm 0.0097$, so $k$-NN does not adapt well to the given learning task (loss function).

Finally, we note that nothing was stopping us from incorporating known invariances into our loss function in KDE via the kernel. For example we could have used a kernel which takes into account local patches of pixels rendering spatial information or jittered kernels which take into account chosen transformations (translations, rotations, and so forth). It may also be useful to add virtual examples to the

output matrix $\mathcal{L}$ before the decomposition step. For an overview of incorporating invariances see [11, Chapter 11] or [12].

## 5   Discussion

We have introduced a kernel method of learning general dependencies. We also gave some first experiments indicating the usefulness of the approach. There are many applications of KDE to explore: problems with complex outputs (natural language parsing, image interpretation/manipulation, ...), applying to special cost functions (e.g ROC scores) and when prior knowledge can be encoded in the outputs.

In terms of further research, we feel there are also still many possibilities to explore in terms of algorithm development. We admit in this work we have a very simplified algorithm for the pre-image part (just choosing the closest image given from the training sample). To make the approach work on more complex problems (where a test output is not so trivially close to a training output) improved pre-image approaches should be applied. Although one can apply techniques such as [10] for vector based pre-images, efficiently finding pre-images for structured objects such as strings is an open problem. Finally, the algorithm should be extended to deal with non-Euclidean loss functions directly, e.g for classification with a general cost matrix. One naive way is to use a distance matrix directly, ignoring the PCA step.

## Footnotes

[1]For instance, assuming the outputs live in $\mathbb{R}^n$, using an RBF kernel, one obtains a loss function $\|\Phi_\ell(\mathbf{y}) - \Phi_\ell(\mathbf{y}')\|^2 = 2 - 2\exp\left(-\|\mathbf{y} - \mathbf{y}'\|^2 / 2\sigma^2\right)$. This is a nonlinear loss function which takes the value 0 if $\mathbf{y}$ and $\mathbf{y}'$ coincide, and 2 if they are maximally different. The rate of increase in between (i.e., the "locality"), is controlled by $\sigma$.

[2] A similar problem, of higher dimensionality, would be to learn the mapping from top half to complete digit.

[3] Note that training a naive regressor on each pixel output independently would not take into account that the combination of pixel outputs should resemble a digit.

## References

[1] N. Cristianini, A. Elisseeff, and J. Shawe-Taylor. On optimizing kernel alignment. Technical Report 2001-087, NeuroCOLT, 2001.

[2] I. Frank and J. Friedman. A Statistical View of Some Chemometrics Regression Tools. *Technometrics*, 35(2):109–147, 1993.

[3] T. Graepel, R. Herbrich, P. Bollmann-Sdorra, and K. Obermayer. Classification on pairwise proximity data. *NIPS*, 11:438–444, 1999.

[4] T. Hastie, R. Tibshirani, and J. Friedman. *The Elements of Statistical Learning*. Springer-Verlag, New York, 2001.

[5] D. Haussler. Convolutional kernels on discrete structures. Technical Report UCSC-CRL-99-10, Computer Science Department, University of California at Santa Cruz, 1999.

[6] J. Li, A. N. Michel, and W. Porod. Analysis and synthesis of a class of neural networks: linear systems operating on a closed hypercube. *IEEE Trans. on Circuits and Systems*, 36(11):1405–22, 1989.

[7] H. Lodhi, C. Saunders, J. Shawe-Taylor, N. Cristianini, and C. Watkins. Text classification using string kernels. *Journal of Machine Learning Research*, 2:419–444, 2002.

[8] S. Mika, G. Rätsch, J. Weston, B. Schölkopf, and K.-R. Müller. Fisher discriminant analysis with kernels. In Y.-H. Hu, J. Larsen, E. Wilson, and S. Douglas, editors, *Neural Networks for Signal Processing IX*, pages 41–48. IEEE, 1999.

[9] C. Saunders, V. Vovk, and A. Gammerman. Ridge regression learning algorithm in dual variables. In J. Shavlik, editor, *Machine Learning Proceedings of the Fifteenth International Conference(ICML '98)*, San Francisco, CA, 1998. Morgan Kaufmann.

[10] B. Schölkopf, S. Mika, C. Burges, P. Knirsch, K.-R. Müller, G. Rätsch, and A. J. Smola. Input space vs. feature space in kernel-based methods. *IEEE-NN*, 10(5):1000–1017, 1999.

[11] B. Schölkopf and A. J. Smola. *Learning with Kernels*. MIT Press, Cambridge, MA, 2002.

[12] V. Vapnik. *Statistical Learning Theory*. John Wiley and Sons, New York, 1998.
